# Correctness of belief propagation in Gaussian graphical models of arbitrary topology

**Yair Weiss**
Computer Science Division
UC Berkeley, 485 Soda Hall
Berkeley, CA 94720-1776
Phone: 510-642-5029
yweiss@cs.berkeley.edu

**William T. Freeman**
Mitsubishi Electric Research Lab
201 Broadway
Cambridge, MA 02139
Phone: 617-621-7527
freeman@merl.com

## Abstract

Local "belief propagation" rules of the sort proposed by Pearl [15] are guaranteed to converge to the correct posterior probabilities in singly connected graphical models. Recently, a number of researchers have empirically demonstrated good performance of "loopy belief propagation"– using these same rules on graphs with loops. Perhaps the most dramatic instance is the near Shannon-limit performance of "Turbo codes", whose decoding algorithm is equivalent to loopy belief propagation.

Except for the case of graphs with a single loop, there has been little theoretical understanding of the performance of loopy propagation. Here we analyze belief propagation in networks with arbitrary topologies when the nodes in the graph describe jointly Gaussian random variables. We give an analytical formula relating the true posterior probabilities with those calculated using loopy propagation. We give sufficient conditions for convergence and show that when belief propagation converges it gives the correct posterior means *for all graph topologies*, not just networks with a single loop.

The related "max-product" belief propagation algorithm finds the maximum posterior probability estimate for singly connected networks. We show that, even for non-Gaussian probability distributions, the convergence points of the max-product algorithm in loopy networks are maxima over a particular large local neighborhood of the posterior probability. These results help clarify the empirical performance results and motivate using the powerful belief propagation algorithm in a broader class of networks.

Problems involving probabilistic belief propagation arise in a wide variety of applications, including error correcting codes, speech recognition and medical diagnosis. If the graph is singly connected, there exist local message-passing schemes to calculate the posterior probability of an unobserved variable given the observed variables. Pearl [15] derived such a scheme for singly connected Bayesian networks and showed that this "belief propagation" algorithm is guaranteed to converge to the correct posterior probabilities (or "beliefs").

Several groups have recently reported excellent experimental results by running algorithms

equivalent to Pearl's algorithm on networks with loops [8, 13, 6]. Perhaps the most dramatic instance of this performance is for "Turbo code" [2] error correcting codes. These codes have been described as "the most exciting and potentially important development in coding theory in many years" [12] and have recently been shown [10, 11] to utilize an algorithm equivalent to belief propagation in a network with loops.

Progress in the analysis of loopy belief propagation has been made for the case of networks with a single loop [17, 18, 4, 1]. For these networks, it can be shown that (1) unless all the compatabilities are deterministic, loopy belief propagation will converge. (2) The difference between the loopy beliefs and the true beliefs is related to the convergence rate of the messages — the faster the convergence the more exact the approximation and (3) If the hidden nodes are binary, then the loopy beliefs and the true beliefs are both maximized by the same assignments, although the confidence in that assignment is wrong for the loopy beliefs.

In this paper we analyze belief propagation in graphs of *arbitrary topology*, for nodes describing jointly Gaussian random variables. We give an exact formula relating the correct marginal posterior probabilities with the ones calculated using loopy belief propagation. We show that if belief propagation converges, then it will give the correct posterior means *for all graph topologies*, not just networks with a single loop. We show that the covariance estimates will generally be incorrect but present a relationship between the error in the covariance estimates and the convergence speed. For Gaussian *or* non-Gaussian variables, we show that the "max-product" algorithm, which calculates the MAP estimate in singly connected networks, only converges to points that are maxima over a particular large neighborhood of the posterior probability of loopy networks.

# 1   Analysis

To simplify the notation, we assume the graphical model has been preprocessed into an undirected graphical model with pairwise potentials. Any graphical model can be converted into this form, and running belief propagation on the pairwise graph is equivalent to running belief propagation on the original graph [18]. We assume each node $x_i$ has a local observation $y_i$. In each iteration of belief propagation, each node $x_i$ sends a message to each neighboring $x_j$ that is based on the messages it received from the other neighbors, its local observation $y_i$ and the pairwise potentials $\Psi_{ij}(x_i, x_j)$ and $\Psi_{ii}(x_i, y_i)$. We assume the message-passing occurs in parallel.

The idea behind the analysis is to build an unwrapped tree. The unwrapped tree is the graphical model which belief propagation is solving exactly when one applies the belief propagation rules in a loopy network [9, 20, 18]. It is constructed by maintaining the same local neighborhood structure as the loopy network but nodes are replicated so there are no loops. The potentials and the observations are replicated from the loopy graph. Figure 1 (a) shows an unwrapped tree for the diamond shaped graph in (b). By construction, the belief at the root node $\bar{x}_1$ is identical to that at node $x_1$ in the loopy graph after four iterations of belief propagation. Each node has a shaded observed node attached to it, omitted here for clarity.

Because the original network represents jointly Gaussian variables, so will the unwrapped tree. Since it is a tree, belief propagation is guaranteed to give the correct answer for the unwrapped graph. We can thus use Gaussian marginalization formulae to calculate the true mean and variances in both the original and the unwrapped networks. In this way, we calculate the accuracy of belief propagation for Gaussian networks of arbitrary topology.

We assume that the joint mean is zero (the means can be added-in later). The joint distri-

Figure 1: **Left:** A Markov network with multiple loops. **Right:** The unwrapped network corresponding to this structure.

bution of $z = \begin{pmatrix} x \\ y \end{pmatrix}$ is given by $P(z) = \alpha e^{-\frac{1}{2}z^T V z}$, where $V = \begin{pmatrix} V_{xx} & V_{xy} \\ V_{yx} & V_{yy} \end{pmatrix}$. It is straightforward to construct the inverse covariance matrix $V$ of the joint Gaussian that describes a given Gaussian graphical model [3].

Writing out the exponent of the joint and completing the square shows that the mean $\mu$ of $x$, given the observations $y$, is given by:

$$V_{xx}\mu = -V_{xy}y, \tag{1}$$

and the covariance matrix $C_{x|y}$ of $x$ given $y$ is: $C_{x|y} = V_{xx}^{-1}$. We will denote by $C_{x_i|y}$ the $i$th row of $C_{x|y}$ so the marginal posterior variance of $x_i$ given the data is $\sigma^2(i) = C_{x_i|y}(i)$.

We will use ˜ for unwrapped quantities. We scan the tree in *breadth first* order and denote by $\tilde{x}$ the vector of values in the hidden nodes of the tree when so scanned. Simlarly, we denote by $\tilde{y}$ the observed nodes scanned in the same order and $\tilde{V}_{xx}, \tilde{V}_{xy}$ the inverse covariance matrices. Since we are scanning in breadth first order the last nodes are the leaf nodes and we denote by $L$ the number of leaf nodes. By the nature of unwrapping, $\tilde{\mu}(1)$ is the mean of the belief at node $x_1$ after $t$ iterations of belief propagation, where $t$ is the number of unwrappings. Similarly $\tilde{\sigma}^2(1) = \tilde{C}_{x_1|y}(1)$ is the variance of the belief at node $x_1$ after $t$ iterations.

Because the data is replicated we can write $\tilde{y} = Oy$ where $O(i,j) = 1$ if $\tilde{y}_i$ is a replica of $y_j$ and 0 otherwise. Since the potentials $\Psi(x_i, y_i)$ are replicated, we can write $\tilde{V}_{xy}O = OV_{xy}$. Since the $\Psi(x_i, x_j)$ are also replicated and all non-leaf $\tilde{x}_i$ have the same connectivity as the corresponding $x_i$, we can write $\tilde{V}_{xx}O = OV_{xx} + E$ where $E$ is zero in all but the last $L$ rows. When these relationships between the loopy and unwrapped inverse covariance matrices are substituted into the loopy and unwrapped versions of equation 1, one obtains the following expression, true for any iteration [19]:

$$\tilde{\mu}(1) = \mu(1) + \tilde{C}_{x_1|y}e \tag{2}$$

where $e$ is a vector that is zero everywhere but the last $L$ components (corresponding to the leaf nodes). Our choice of the node for the root of the tree is arbitrary, so this applies to all nodes of the loopy network. This formula relates, for any node of a network with loops, the means calculated at each iteration by belief propagation with the true posterior means.

Similarly when the relationship between the loopy and unwrapped inverse covariance matrices is substituted into the loopy and unwrapped definitions of $C_{x|y}$ we can relate the

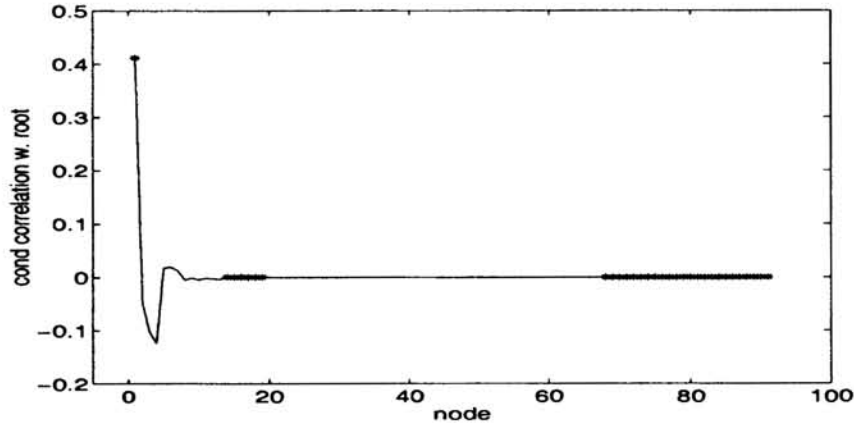

Figure 2: The conditional correlation between the root node and all other nodes in the unwrapped tree of Fig. 1 after eight iterations. Potentials were chosen randomly. Nodes are presented in breadth first order so the last elements are the correlations between the root node and the leaf nodes. We show that if this correlation goes to zero, belief propagation converges and the loopy means are exact. Symbols plotted with a star denote correlations with nodes that correspond to the node $x_1$ in the loopy graph. The sum of these correlations gives the correct variance of node $x_1$ while loopy propagation uses only the first correlation.

marginalized covariances calculated by belief propagation to the true ones [19]:

$$\tilde{\sigma}^2(1) = \sigma^2(1) + \tilde{C}_{x_1|y}e_1 - \tilde{C}_{x_1|y}e_2 \tag{3}$$

where $e_1$ is a vector that is zero everywhere but the last $L$ components while $e_2$ is equal to 1 for all nodes in the unwrapped tree that are replicas of $x_1$ except for $\tilde{x}_1$. All other components of $e_2$ are zero,

Figure 2 shows $\tilde{C}_{x_1|y}$ for the diamond network in Fig. 1. We generated random potential functions and observations and calculated the conditional correlations in the unwrapped tree. Note that the conditional correlation decreases with distance in the tree — we are scanning in breadth first order so the last $L$ components correspond to the leaf nodes. As the number of iterations of loopy propagation is increased the size of the unwrapped tree increases and the conditional correlation between the leaf nodes and the root node decreases.

From equations 2–3 it is clear that if the conditional correlation between the leaf nodes and the root nodes are zero for all sufficiently large unwrappings then (1) belief propagation converges (2) the means are exact and (3) the variances may be incorrect. In practice the conditional correlations will not actually be equal to zero for any finite unwrapping. In [19] we give a more precise statement: if the conditional correlation of the root node and the leaf nodes decreases rapidly enough then (1) belief propagation converges (2) the means are exact and (3) the variances may be incorrect. We also show sufficient conditions on the potentials $\Psi(x_i, x_j)$ for the correlation to decrease rapidly enough: the rate at which the correlation decreases is determined by the ratio of off-diagonal and diagonal components in the quadratic form defining the potentials [19].

How wrong will the variances be? The term $\tilde{C}_{x_1|y}e_2$ in equation 3 is simply the sum of many components of $\tilde{C}_{x_1|y}$. Figure 2 shows these components. The correct variance is the sum of all the components while the belief propagation variance approximates this sum with the first (and dominant) term. Whenever there is a positive correlation between the root node and other replicas of $x_1$ the loopy variance is strictly less than the true variance — the loopy estimate is overconfident.

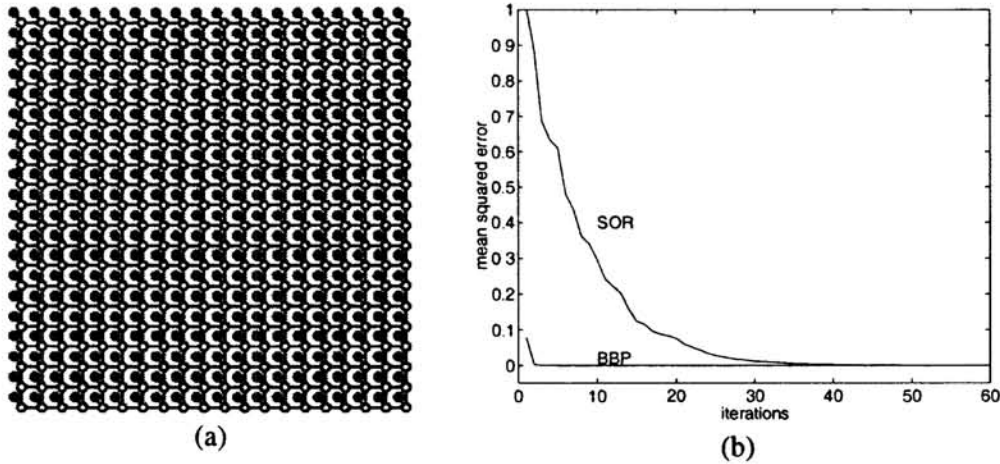

Figure 3: (a) $25 \times 25$ graphical model for simulation. The unobserved nodes (unfilled) were connected to their four nearest neighbors and to an observation node (filled). (b) The error of the estimates of loopy propagation and successive over-relaxation (SOR) as a function of iteration. Note that belief propagation converges much faster than SOR.

Note that when the conditional correlation decreases rapidly to zero two things happen. First, the convergence is faster (because $\tilde{C}_{x_1|y}e_1$ approaches zero faster). Second, the approximation error of the variances is smaller (because $\tilde{C}_{x_1|y}e_2$ is smaller). Thus we have shown, as in the single loop case, quick convergence is correlated with good approximation.

## 2   Simulations

We ran belief propagation on the $25 \times 25$ $2D$ grid of Fig. 3 a. The joint probability was:

$$P(x, y) = exp(-\sum_{ij} w_{ij}(x_i - x_j)^2 - \sum_i w_{ii}(x_i - y_i)^2) \qquad (4)$$

where $w_{ij} = 0$ if nodes $x_i, x_j$ are not neighbors and 0.01 otherwise and $w_{ii}$ was randomly selected to be 0 or 1 for all $i$ with probability of 1 set to 0.2. The observations $y_i$ were chosen randomly. This problem corresponds to an approximation problem from sparse data where only 20% of the points are visible.

We found the exact posterior by solving equation 1. We also ran belief propagation and found that when it converged, the calculated means were identical to the true means up to machine precision. Also, as predicted by the theory, the calculated variances were too small — the belief propagation estimate was overconfident.

In many applications, the solution of equation 1 by matrix inversion is intractable and iterative methods are used. Figure 3 compares the error in the means as a function of iterations for loopy propagation and successive-over-relaxation (SOR), considered one of the best relaxation methods [16]. Note that after essentially five iterations loopy propagation gives the right answer while SOR requires many more. As expected by the fast convergence, the approximation error in the variances was quite small. The median error was 0.018. For comparison the true variances ranged from 0.01 to 0.94 with a mean of 0.322. Also, the nodes for which the approximation error was worse were indeed the nodes that converged slower.

## 3   Discussion

Independently, two other groups have recently analyzed special cases of Gaussian graphical models. Frey [7] analyzed the graphical model corresponding to factor analysis and gave conditions for the existence of a stable fixed-point. Rusmevichientong and Van Roy [14] analyzed a graphical model with the topology of turbo decoding but a Gaussian joint density. For this specific graph they gave sufficient conditions for convergence and showed that the means are exact.

Our main interest in the Gaussian case is to understand the performance of belief propagation in general networks with multiple loops. We are struck by the similarity of our results for Gaussians in arbitrary networks and the results for single loops of arbitrary distributions [18]. First, in single loop networks with binary nodes, loopy belief at a node and the true belief at a node are maximized by the same assignment while the confidence in that assignment is incorrect. In Gaussian networks with multiple loops, the mean at each node is correct but the confidence around that mean may be incorrect. Second, for both single-loop and Gaussian networks, fast belief propagation convergence correlates with accurate beliefs. Third, in both Gaussians and discrete valued single loop networks, the statistical dependence between root and leaf nodes governs the convergence rate and accuracy.

The two models are quite different. Mean field approximations are exact for Gaussian MRFs while they work poorly in sparsely connected discrete networks with a single loop. The results for the Gaussian and single-loop cases lead us to believe that similar results may hold for a larger class of networks.

Can our analysis be extended to non-Gaussian distributions? The basic idea applies to arbitrary graphs and arbitrary potentials: belief propagation is performing exact inference on a tree that has the same local neighbor structure as the loopy graph. However, the linear algebra that we used to calculate exact expressions for the error in belief propagation at any iteration holds only for Gaussian variables.

We have used a similar approach to analyze the related "max-product" belief propagation algorithm on arbitrary graphs with arbitrary distributions [5] (both discrete and continuous valued nodes). We show that if the max-product algorithm converges, the max-product assignment has greater posterior probability then any assignment in a particular large region around that assignment. While this is a weaker condition than a global maximum, it is much stronger than a simple local maximum of the posterior probability.

The sum-product and max-product belief propagation algorithms are fast and parallelizable. Due to the well known hardness of probabilistic inference in graphical models, belief propagation will obviously not work for arbitrary networks and distributions. Nevertheless, a growing body of empirical evidence shows its success in many networks with loops. Our results justify applying belief propagation in certain networks with multiple loops. This may enable fast, approximate probabilistic inference in a range of new applications.

## References

[1] S.M. Aji, G.B. Horn, and R.J. McEliece. On the convergence of iterative decoding on graphs with a single cycle. In *Proc. 1998 ISIT*, 1998.

[2] C. Berrou, A. Glavieux, and P. Thitimajshima. Near Shannon limit error-correcting coding and decoding: Turbo codes. In *Proc. IEEE International Communications Conference '93*, 1993.

[3] R. Cowell. Advanced inference in Bayesian networks. In M.I. Jordan, editor, *Learning in Graphical Models*. MIT Press, 1998.

[4] G.D. Forney, F.R. Kschischang, and B. Marcus. Iterative decoding of tail-biting trellisses. preprint presented at 1998 Information Theory Workshop in San Diego, 1998.

[5] W. T. Freeman and Y. Weiss. On the fixed points of the max-product algorithm. Technical Report 99–39, MERL, 201 Broadway, Cambridge, MA 02139, 1999.

[6] W.T. Freeman and E.C. Pasztor. Learning to estimate scenes from images. In M.S. Kearns, S.A. Solla, and D.A. Cohn, editors, *Adv. Neural Information Processing Systems 11*. MIT Press, 1999.

[7] B.J. Frey. Turbo factor analysis. In *Adv. Neural Information Processing Systems 12*. 2000. to appear.

[8] Brendan J. Frey. *Bayesian Networks for Pattern Classification, Data Compression and Channel Coding*. MIT Press, 1998.

[9] R.G. Gallager. *Low Density Parity Check Codes*. MIT Press, 1963.

[10] F. R. Kschischang and B. J. Frey. Iterative decoding of compound codes by probability propagation in graphical models. *IEEE Journal on Selected Areas in Communication*, 16(2):219–230, 1998.

[11] R.J. McEliece, D.J.C. MackKay, and J.F. Cheng. Turbo decoding as as an instance of Pearl's 'belief propagation' algorithm. *IEEE Journal on Selected Areas in Communication*, 16(2):140–152, 1998.

[12] R.J. McEliece, E. Rodemich, and J.F. Cheng. The Turbo decision algorithm. In *Proc. 33rd Allerton Conference on Communications, Control and Computing*, pages 366–379, Monticello, IL, 1995.

[13] K.P. Murphy, Y. Weiss, and M.I. Jordan. Loopy belief propagation for approximate inference: an empirical study. In *Proceedings of Uncertainty in AI*, 1999.

[14] Rusmevichientong P. and Van Roy B. An analysis of Turbo decoding with Gaussian densities. In *Adv. Neural Information Processing Systems 12*. 2000. to appear.

[15] Judea Pearl. *Probabilistic Reasoning in Intelligent Systems: Networks of Plausible Inference*. Morgan Kaufmann, 1988.

[16] Gilbert Strang. *Introduction to Applied Mathematics*. Wellesley-Cambridge, 1986.

[17] Y. Weiss. Belief propagation and revision in networks with loops. Technical Report 1616, MIT AI lab, 1997.

[18] Y. Weiss. Correctness of local probability propagation in graphical models with loops. *Neural Computation*, to appear, 2000.

[19] Y. Weiss and W. T. Freeman. Loopy propagation gives the correct posterior means for Gaussians. Technical Report UCB.CSD-99-1046, Berkeley Computer Science Dept., 1999. www.cs.berkeley.edu yweiss/.

[20] N. Wiberg. *Codes and decoding on general graphs*. PhD thesis, Department of Electrical Engineering, U. Linkoping, Sweden, 1996.